# Phase-coupling in Two-Dimensional Networks of Interacting Oscillators

Ernst Niebur, Daniel M. Kammen, Christof Koch,
Daniel Ruderman[1] & Heinz G. Schuster[2]
Computation and Neural Systems
Caltech 216-76
Pasadena, CA 91125

## ABSTRACT

Coherent oscillatory activity in large networks of biological or artificial neural units may be a useful mechanism for coding information pertaining to a single perceptual object or for detailing regularities within a data set. We consider the dynamics of a large array of simple coupled oscillators under a variety of connection schemes. Of particular interest is the rapid and robust phase-locking that results from a "sparse" scheme where each oscillator is strongly coupled to a tiny, randomly selected, subset of its neighbors.

## 1   INTRODUCTION

Networks of interacting oscillators provide an excellent model for numerous physical processes ranging from the behavior of magnetic materials to models of atmospheric dynamics to the activity of populations of neurons in a variety of cortical locations. Particularly prominent in the neurophysiological data are the $40-60\ Hz$ oscillations that have long been reported in the rat and rabbit olfactory bulb and cortex on the basis of single-and multi-unit recordings as well as EEG activity (Freeman, 1978). In addition, periodicities in eye movement reaction times (Pöppel and Logothetis, 1986), as well as oscillations in the auditory evoked potential in response to single click or a series of clicks (Madler and Pöppel, 1987) all support a $30-50\ Hz$ framework for aspects of cortical activity. Two groups (Eckhorn *et al.*, 1988, Gray

and Singer, 1989; Gray *et al.*, 1989) have recently reported highly synchronized, stimulus specific oscillations in the $35 - 85\ Hz$ range in areas 17, 18 and PMLS of anesthetized as well as awake cats. Neurons with similar orientation tuning up to $7\ mm$ apart show phase-locked oscillations with a phase shift of less than $1\ msec$ that have been proposed to play a role in the coding of visual information (Crick and Koch, 1990, Niebur *et al.* 1991).

The complexity of networks of even relatively simple neuronal units – let alone "real" cortical cells – warrants a systematic investigation of the behavior of two dimensional systems. To address this question we begin with a network of mathematically simple limit-cycle oscillators. While the dynamics of pairs of oscillators are well understood (Sakaguchi, *et al.* 1988, Schuster and Wagner, 1990a,b), this is not the case for large networks with nontrivial connection schemes. Of general interest is the phase-coupling that results in networks of oscillators with different coupling schemes. We will summarize some generic features of simple nearest-neighbor coupled models, models where each oscillator receives input from a large neighborhood, and of "sparse" connection geometries where each cell is connected to only a tiny fraction of the units in its neighborhood, but with large coupling strength. The numerical work was performed on a CM-2 Connection Machine and involved 16,384 oscillators in a 128 by 128 square grid.

## 2   The Model

The basic unit in our networks is an oscillator whose phase $\theta_{ij}$ is $2\pi$ periodic and which has the intrinsic frequency $\omega_{ij}$. The dynamics of an isolated oscillator are described by:

$$\frac{d\theta_{ij}}{dt} = \omega_{ij}. \tag{1}$$

The influence of the network can be expressed as an additional interaction term,

$$\frac{d\theta_{ij}}{dt} = \omega_{ij} + f_{ij}(\theta_0, \theta_1, ... \theta_n). \tag{2}$$

The coupling function, $f_{ij}$ we used is expressed as the sum of terms, each one consisting of the product of a coupling strength and the sine of a phase difference (see below, eq. 3). The sinusoidal form of the interaction is, of course, linear for small differences.

This system, and numerous variants, has received a considerable amount of attention from solid state physicists (see, e.g. Kosterlitz and Thouless 1973, and Sakaguchi *et al.* 1988), although primarily in the limit of $t \to \infty$. With an interest in the possible role of networks of oscillators in the parsing or segregating of incident signals in nervous systems, we will concentrate on short time, non-equilibrium, properties.

We shall confine ourselves to two generic network configurations described by

$$\frac{d\theta_{ij}}{dt} = \omega_{ij} + \alpha \sum_{kl} J_{ij,kl} sin(\theta_{ij} - \theta_{kl}), \tag{3}$$

where $\alpha$ designates the global strength of the interaction, and the geometry of the interactions is incorporated in $J_{ij,kl}$.

The networks are all defined on a square grid and they are characterized as follows:

**1: Gaussian Connections.** The cells are connected to every oscillator within a specified neighborhood with Gaussian weighted connections. Hence,

$$J_{ij,kl} = \frac{1}{2\pi\sigma} exp\left(\frac{(i-k)^2 + (j-l)^2}{2\sigma^2}\right). \tag{4}$$

We truncate this function at $2\sigma$, i.e. $J_{ij,kl} = 0$ if $(i-k)^2 + (j-l)^2 \geq (2\sigma)^2$. While the connectivity in the nearest neighbor case is 4, the connectivity is significantly higher for the Gaussian connection schemes: Already $\sigma = 2$ yields 28 connections per cell, and the largest network we studied, with $\sigma = 6$, results in 372 connections per cell.

**2: Sparse Gaussian Connections.** In this scheme we no longer require symmetric connections, or that the connection pattern is identical from unit to unit. A given cell is connected to a fixed *number*, $n$, of neighboring cells, with the probability of a given connection determined by

$$\mathcal{P}_{ij,kl} = \frac{1}{2\pi\sigma} exp\left(\frac{(i-k)^2 + (j-l)^2}{2\sigma^2}\right). \tag{5}$$

$J_{ij,kl}$ is unity with probability $\mathcal{P}_{ij,kl}$ and zero otherwise. This connection scheme is constructed by drawing for each lattice site $n$ coordinate pairs from a Gaussian distribution, and use these as the indices of the cells that are connected with the oscillator at location $(i,j)$. Therefore, the probability of making a connection decreases with distance. If a connection is made, however, the weight is the same as for all other connections. We typically used $n = 5$, and in all cases $2 \leq n \leq 10$.

For all networks, the sum of the weights of all connections with a given oscillator $i, j$ was conserved and chosen as $\alpha \sum_{kl} J_{ij,kl} = 10 * \overline{\omega}$, where $\overline{\omega}$ is the average frequency of all $N$ oscillators in the system, $\overline{\omega} = \frac{1}{N}\sum_{ij}\omega_{ij}$. By this procedure, the total impact of the interaction term is identical in all cases.

## 3   RESULTS

Perhaps the most basic, and most revealing, comparison of the behavior of the models introduced above is the two-point correlation function of phase-coupling, which is defined as

$$C(R,t) = < cos\left[\theta_{ij}(t) - \theta_{kl}(t)\right] >, \tag{6}$$

where $R$ is defined as the separation between a pair of cells, $R = |r_{ij} - r_{kl}|$. We compute and then average $C(R,t)$ over 10,000 pairs of oscillators separated by $R$ in the array. In all cases, the frequencies $\omega_{ij}$ are chosen randomly, with a Gaussian distribution with mean 0.5 and variance 1. In Figure 1 we plot $C(R,t)$ for separations

of $R = 20$, 30, 40, 50, 6, and 70 oscillators. Time is measured in oscillation periods of the mean oscillator frequency, $\bar{\omega}$. At $t = 0$, phases are distributed randomly between 0 and $2\pi$ with a uniform distribution. The case of Gaussian connectivity with $\sigma = 6$ and hence 372 connection per cell is seen in Figure 1(a), and the sparse connectivity scheme with $\sigma = 6$ and $n = 5$ is presented in Figure 1(b). The most striking difference is that correlation levels of over 0.9 are rapidly achieved in the sparse scheme for all cases, even for separations of 70 oscillators (plotted as asterisks, *), while there are clear separation-dependent differences in the phase-locking behavior of the Gaussian model. In fact, even after $t = 10$ there is no significant locking over the longer distances of $R = 50, 60$, or 70 units. For local connectivity schemes, like Gaussian connectivity with $\sigma = 2$ or nearest neighbors connections, no long-range order evolves even at larger times (data not shown).

Data in Fig. 1 were computed with a uniform phase distribution for $t = 0$. An interesting and robust feature of the dynamics emerges when the influence of different types of initial phase distributions are examined. In Figure 2 we plot the probability distribution of phases at different early times. In Figure 2(a) the distribution of phases is plotted at $t = 0$ (diamonds), $t = 0.2$ ("plus signs, +) and at $t = 0.4$ (squares) for the sparse scheme with a uniform initial distribution. In Figure 2(b), the evolution of a Gaussian initial distribution centered at $\theta = \pi$ of the phases is plotted. Note the slight curve in the distribution at $t = 0$, indicating that the Gaussian initial seeding is rather slight (variance $\sigma = 2\pi$). Remarkably, however, this has a dramatic impact on the phase-locking as after two-tenth of an average cycle time ("plus" signs) there is already a pronounced peak in the distribution. At $t = 0.4$ (squares) the system that started with the uniform distribution begins to only exhibit a slight increase in the phase-correlation while the system with Gaussian distributed initial phases is strongly peaked with virtually no probability of encountering phase values that differ significantly from the mean.

## 4    DISCUSSION

The power of the sparse connection scheme to rapidly generate phase-locking throughout the network that is equivalent, or superior, to that of the massively interconnected Gaussian scheme highlights a trade-off in network dynamics: massive averaging versus strong, long-range, connections. With $n = 5$, the sparse scheme effectively "tiles" a two-dimensional lattice and tightly phase-locks oscillators even at opposite corners of the array. Similar results are obtained even with $n = 2$ (data not shown).

In many ways the Gaussian and sparse geometries reperent opposing avenues to achieve global coherence: exhaustive local coupling or distributed, but powerful long-range coupling. The amount of wiring necessary to implement these schemes is, however, radically different.

## Acknowledgement

EN is supported by the Swiss National Science Foundation through Grant No. 8220-25941. DMK is a recipient of a Weizman Postdoctoral Fellowship. CK acknowledges support from the Air Force Office of Scientific Research, a NSF Presidential Young Investigator Award and from the James S. McDonnell Foundation. HGS is supported by the Volkswagen Foundation.

## Footnotes

[1]Permanent address: Department of Physics, University of California, Berkeley, CA 94720

[2]Permanent address: Institut für Theoretische Physik, Universität Kiel, 2300 Kiel 1, Germany.

## References

Crick, F. and Koch, C. 1990. Towards a neurobiological theory of consciousness. *Seminars Neurosci.*, **2**, 263 - 275.

Eckhorn, R., Bauer, R., Jordan, W., Brosch, M., Kruse, W., Munk, M. and Reitboeck, H. J. 1988. Coherent oscillations: A mechanism of feature linking in the visual cortex? *Biol. Cybern.*, **60**, 121 - 130.

Freeman, W. J. 1978. Spatial properties of an EEG event in the olfactory bulb and cortex. *Elect. Clin. Neurophys.*, **44**, 586 - 605.

Gray, C. M., König, P., Engel, A. K. and Singer, W. 1989. Oscillatory responses in cat visual cortex exhibit inter-columnar synchronization which reflects global stimulus properties. *Nature*, **338**, 334 -337.

Kosterlitz, J. M. and Thouless, D. J. 1973. Ordering, metastability and phase transitions in two-dimensional systems. *J. Physics C.*, **6**, 1181 - 1203.

Madler, C. and Pöppel, E. 1987. Auditory evoked potentials indicate the loss of neuronal oscillations during general anaesthesia. *Naturwissenschaften*, **74**, 42 - 43.

Niebur, E., Kammen, D. M., and Koch, C. 1991. Phase-locking in 1-D and 2-D networks of oscillating neurons. In *Nonlinear dynamics and neuronal networks*, Singer, W., and Schuster, H. G. (eds.). VCH Verlag: Weinheim, FRG.

Pöppel, E. and Logothetis, N. 1986. Neuronal oscillations in the human brain. *Naturwissenschaften*, **73**, 267 - 268.

Sakaguchi, H., Shinomoto, S. and Kuramoto, Y. 1988. Mutual entrainment in oscillator lattices with nonvariational type interaction. *Prog. Theor. Phys.*, **79**, 1069 - 1079.

Schuster, H. G. and Wagner, P. 1990a. A model for neuronal oscillations in the visual cortex 1: Mean-field theory and derivation of phase equations. *Biological Cybernetics*, **64**, 77 - 82.

Schuster, H. G. and Wagner, P. 1990b. A model for neuronal oscillations in the visual cortex 2: Phase description of feature dependant synchronization. *Biological Cybernetics*, **64**, 83.

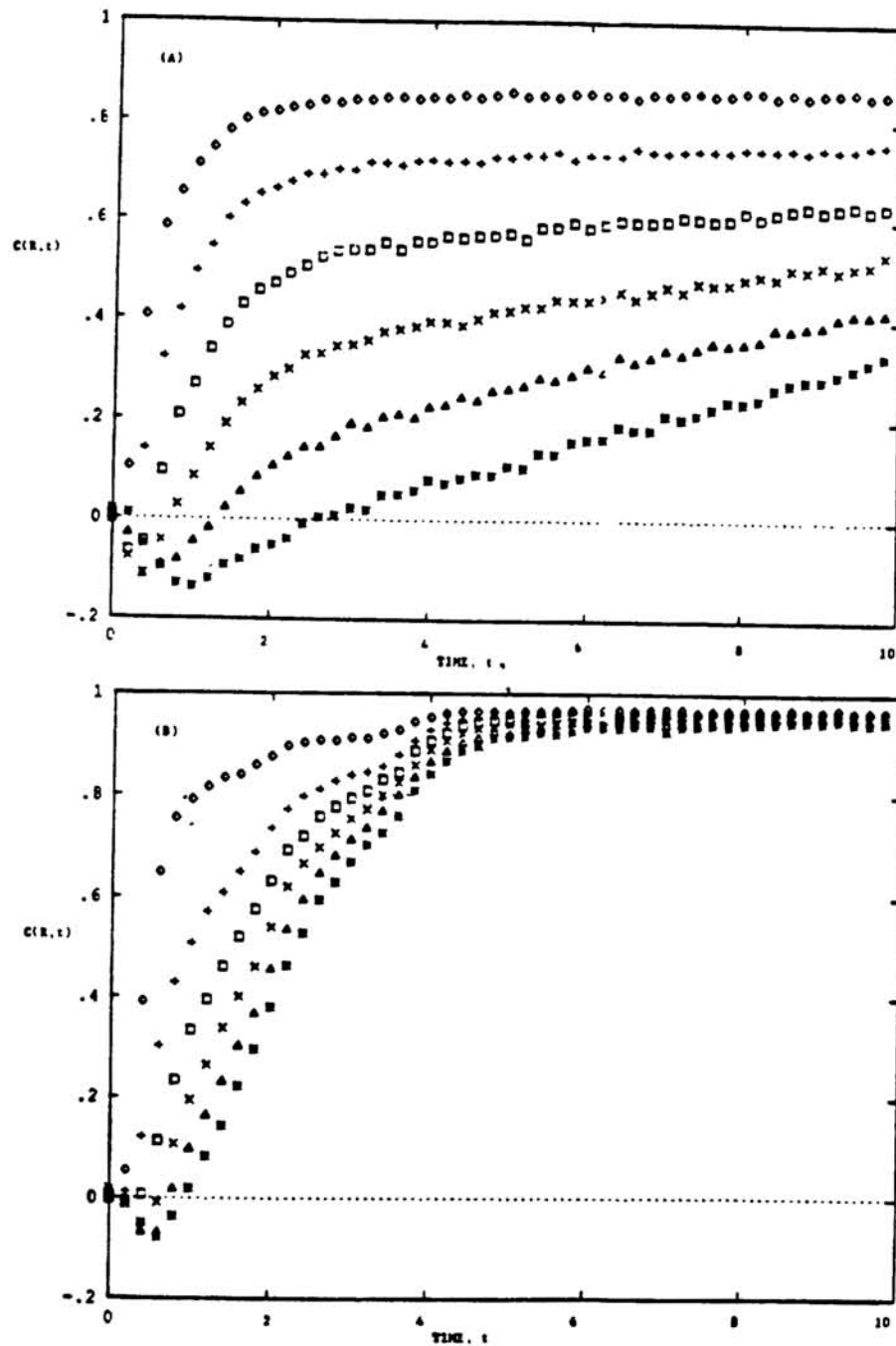

**Figure 1:** Two-point correlation functions, $C(R, t)$, for various separations, R, in (a) the $\sigma = 6$ Gaussian scheme with 372 connections per cell and (b) the sparse connection scheme with $\sigma = 6$ and $n = 5$ connections per cell. Separations of $R = 20$ (diamonds), $R = 30$ ("plus" signs, +), $R = 40$ (squares), $R = 50$ (crosses, ×), $R = 60$ (triangles), and $R = 70$ (asterisks, *) are shown. Note the rapid locking for all lengths in the sparse scheme (b) while the Gaussian scheme (a) appears far more "diffusive," with progressively poorer and slower locking as $R$ increases.

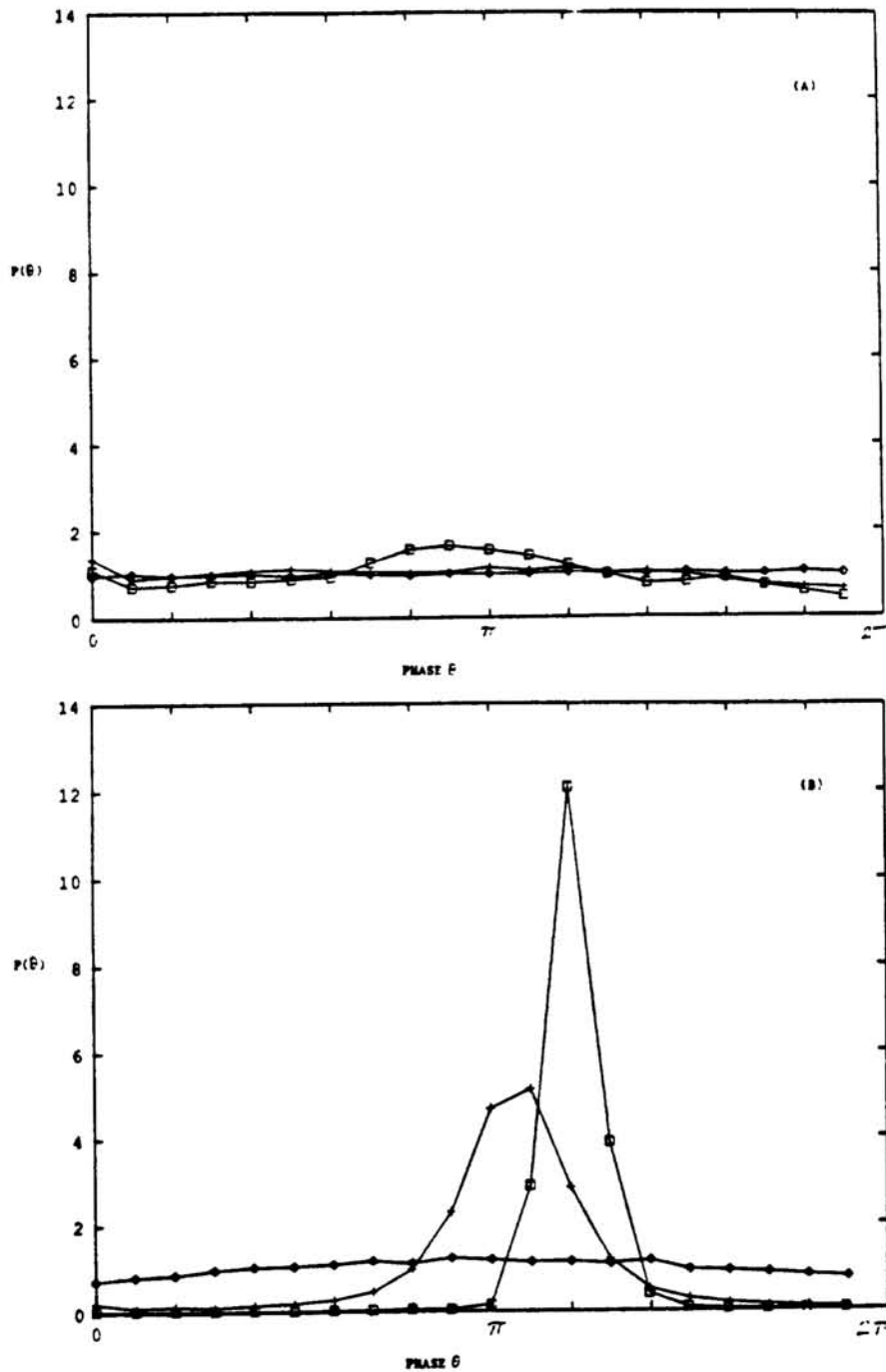

**Figure 2:** Snapshots of the distribution of phases in the sparse scheme ($n = 5, \sigma = 6$) when the system begins from (a) uniform and (b) a Gaussian "biased" initial distribution. The figures show the probability $P(\theta)$ to find a phase between $\theta$ and $\theta + d\theta$ (bin size $\pi/10$). At $t = 0$, the distribution is flat (a) or very slightly curved (b); see text. The difference in the time evolution can clearly be seen in the state of the system after $t = 0.2$ ("plus" signs, $+$) and $t = 0.4$ (squares).